# Deep Neural Networks Segment Neuronal Membranes in Electron Microscopy Images

**Dan C. Cireşan**[*]
IDSIA
USI-SUPSI
Lugano 6900
dan@idsia.ch

**Alessandro Giusti**
IDSIA
USI-SUPSI
Lugano 6900
alessandrog@idsia.ch

**Luca M. Gambardella**
IDSIA
USI-SUPSI
Lugano 6900
luca@idsia.ch

**Jürgen Schmidhuber**
IDSIA
USI-SUPSI
Lugano 6900
juergen@idsia.ch

## Abstract

We address a central problem of neuroanatomy, namely, the automatic segmentation of neuronal structures depicted in stacks of electron microscopy (EM) images. This is necessary to efficiently map 3D brain structure and connectivity. To segment *biological* neuron membranes, we use a special type of deep *artificial* neural network as a pixel classifier. The label of each pixel (membrane or non-membrane) is predicted from raw pixel values in a square window centered on it. The input layer maps each window pixel to a neuron. It is followed by a succession of convolutional and max-pooling layers which preserve 2D information and extract features with increasing levels of abstraction. The output layer produces a calibrated probability for each class. The classifier is trained by plain gradient descent on a $512 \times 512 \times 30$ stack with known ground truth, and tested on a stack of the same size (ground truth unknown to the authors) by the organizers of the ISBI 2012 EM Segmentation Challenge. Even without problem-specific post-processing, our approach outperforms competing techniques by a large margin in all three considered metrics, i.e. *rand error*, *warping error* and *pixel error*. For pixel error, our approach is the only one outperforming a second human observer.

## 1 Introduction

How is the brain structured? The recent field of connectomics [2] is developing high-throughput techniques for mapping connections in nervous systems, one of the most important and ambitious goals of neuroanatomy. The main tool for studying connections at the neuron level is serial-section Transmitted Electron Microscopy (ssTEM), resolving individual neurons and their shapes. After preparation, a sample of neural tissue is typically sectioned into 50-nanometer slices; each slice is then recorded as a 2D grayscale image with a pixel size of about $4 \times 4$ nanometers (see Figure 1).

The visual complexity of the resulting stacks makes them hard to handle. Reliable automated segmentation of neuronal structures in ssTEM stacks so far has been infeasible. A solution of this problem, however, is essential for any automated pipeline reconstructing and mapping neural connections in 3D. Recent advances in automated sample preparation and imaging make this increas-

---

[*]webpage: http://www.idsia.ch/~ciresan

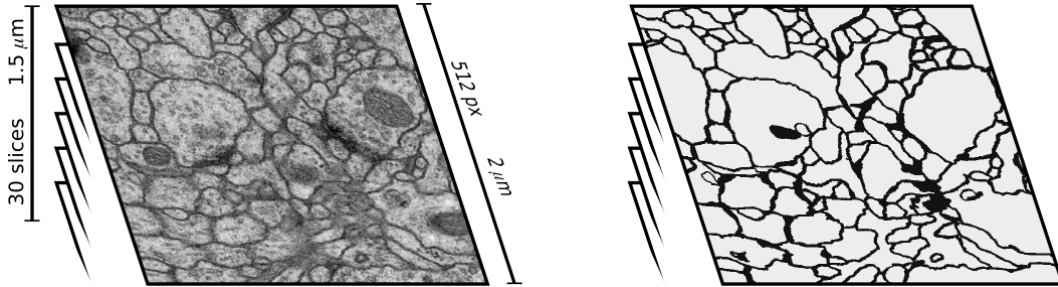

Figure 1: Left: the training stack (one slice shown). Right: corresponding ground truth; black lines denote neuron membranes. Note complexity of image appearance.

ingly urgent, as they enable acquisition of huge datasets [6, 21], whose manual analysis is simply unfeasible.

Our solution is based on a Deep Neural Network (DNN) [12, 13] used as a pixel classifier. The network computes the probability of a pixel being a membrane, using as input the image intensities in a square window centered on the pixel itself. An image is then segmented by classifying all of its pixels. The DNN is trained on a different stack with similar characteristics, in which membranes were manually annotated.

DNN are inspired by convolutional neural networks introduced in 1980 [16], improved in the 1990s [25], refined and simplified in the 2000s [5, 33], and brought to their full potential by making them both large and deep [12, 13]. Lately, DNN proved their efficiency on data sets extending from handwritten digits (MNIST) [10, 12], handwritten characters [11] to 3D toys (NORB) [13] and faces [35]. Training huge nets requires months or even years on CPUs, where high data transfer latency prevented multi-threading code from saving the situation. Our fast GPU implementation [10, 12] overcomes this problem, speeding up single-threaded CPU code by up to two orders of magnitude.

Many other types of learning classifiers have been applied to segmentation of TEM images, where different structures are not easily characterized by intensity differences, and structure boundaries are not correlated with high image gradients, due to noise and many confounding micro-structures. In most binary segmentation problems, classifiers are used to compute one or both of the following probabilities: *(a)* probability of a pixel belonging to each class; *(b)* probability of a boundary dividing two adjacent pixels. Segmentation through graph cuts [7] uses *(a)* as the unary term, and *(b)* as the binary term. Some use an additional term to account for the expected geometry of neuron membranes[23].

We compute pixel probabilities only (point *(a)* above), and directly obtain a segmentation by mild smoothing and thresholding, without using graph cuts. Our main contribution lies therefore in the classifier itself. Others have used off-the-shelf random forest classifiers to compute unary terms of neuron membranes [22], or SVMs to compute both unary and binary terms for segmenting mitochondria [28, 27]. The former approach uses haar-like features and texture histograms computed on a small region around the pixel of interest, whereas the latter uses sophisticated rotational [17] and ray [34] features computed on superpixels [3]. Feature selection mirrors the researcher's expectation of which characteristics of the image are relevant for classification, and has a large impact on classification accuracy. In our approach, we bypass such problems, using raw pixel values as inputs. Due to their convolutional structure, the first layers of the network automatically learn to compute meaningful features during training.

The main contribution of the paper is a practical state-of-the-art segmentation method for neuron membranes in ssTEM data, described in Section 2. It outperforms existing methods as validated in Section 3. The contribution is particularly meaningful because our approach does not rely on problem-specific postprocessing: fruitful application to different biomedical segmentation problems is therefore likely.

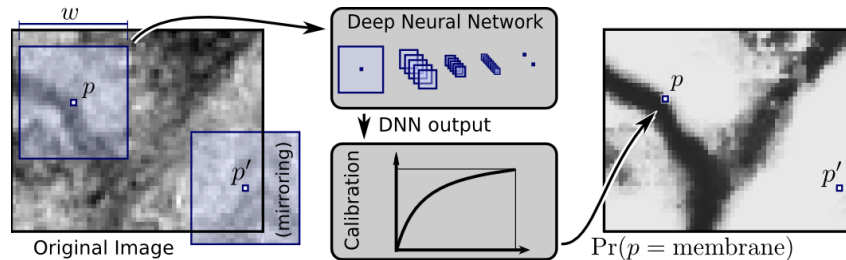

Figure 2: Overview of our approach (see text).

## 2 Methods

For each pixel we consider two possible classes, *membrane* and *non-membrane*. The DNN classifier (Section 2.1) computes the probability of a pixel $p$ being of the former class, using as input the raw intensity values of a square window centered on $p$ with an edge of $w$ pixels—$w$ being an odd number to enforce symmetry. When a pixel is close to the image border, its window will include pixels outside the image boundaries; such pixels are synthesized by mirroring the pixels in the actual image across the boundary (see Figure 2).

The classifier is first trained using the provided training images (Section 2.2). After training, to segment a test image, the classifier is applied to all of its pixels, thus generating a map of *membrane* probabilities—i.e., a new real-valued image the size of the input image. Binary membrane segmentation is obtained by mild postprocessing techniques discussed in Section 2.3, followed by thresholding.

### 2.1 DNN architecture

A DNN [13] consists of a succession of convolutional, max-pooling and fully connected layers. It is a general, hierarchical feature extractor that maps raw pixel intensities of the input image into a feature vector to be classified by several fully connected layers. All adjustable parameters are jointly optimized through minimization of the misclassification error over the training set.

Each **convolutional layer** performs a 2D convolution of its input maps with a square filter. The activations of the output maps are obtained by summing the convolutional responses which are passed through a nonlinear activation function.

The biggest architectural difference between the our DNN and earlier CNN [25] are **max-pooling layers** [30, 32, 31] instead of sub-sampling layers. Their outputs are given by the maximum activation over non-overlapping square regions. Max-pooling are fixed, non-trainable layers which select the most promising features. The DNN also have many more maps per layer, and thus many more connections and weights.

After 1 to 4 stages of convolutional and max-pooling layers several **fully connected layers** further combine the outputs into a 1D feature vector. The output layer is always a fully connected layer with one neuron per class (two in our case). Using a softmax activation function for the last layer guarantees that each neuron's output activation can be interpreted as the probability of a particular input image belonging to that class.

### 2.2 Training

To train the classifier, we use all available slices of the training stack, i.e., 30 images with a $512 \times 512$ resolution. For each slice, we use all *membrane* pixels as positive examples (on average, about $50000$), and the same amount of pixels randomly sampled (without repetitions) among all *non-membrane* pixels. This amounts to 3 million training examples in total, in which both classes are equally represented.

As is often the case in TEM images—but not in other modalities such as phase-contrast microscopy—the appearance of structures is not affected by their orientation. We take advantage of

this property, and synthetically augment the training set at the beginning of each epoch by randomly mirroring each training instance, and/or rotating it by $\pm 90°$.

## 2.3 Postprocessing of network outputs

Because each class is equally represented in the training set but not in the testing data, the network outputs cannot be directly interpreted as probability values; instead, they tend to severely overestimate the membrane probability. To fix this issue, a polynomial function post-processor is applied to the network outputs.

To compute its coefficients, a network $\mathbf{N}$ is trained on 20 slices of the training volume $\mathbf{T}_{\text{train}}$ and tested on the remaining 10 slices of the same volume ($\mathbf{T}_{\text{test}}$, for which ground truth is available). We compare all outputs obtained on $\mathbf{T}_{\text{test}}$ (a total of 2.6 million instances) to ground truth, to compute the transformation relating the network output value and the actual probability of being a membrane; for example, we measure that, among all pixels of $\mathbf{T}_{\text{test}}$ which were classified by $\mathbf{N}$ as having a $50\%$ probability of being *membrane*, only about $18\%$ have in fact such a ground truth label; the reason being the different prevalence of membrane instances in $\mathbf{T}_{\text{train}}$ (i.e. $50\%$) and in $\mathbf{T}_{\text{test}}$ (roughly $20\%$). The resulting function is well approximated by a monotone cubic polynomial, whose coefficients are computed by least-squares fitting. The same function is then used to calibrate the outputs of all trained networks.

After calibration (a grayscale transformation in image processing terms), network outputs are spatially smoothed by a 2-pixel-radius median filter. This results in regularized of membrane boundaries after thresholding.

## 2.4 Foveation and nonuniform sampling

We experimented with two related techniques for improving the network performance by manipulating its input data, namely *foveation* and *nonuniform sampling* (see Figure 3).

*Foveation* is inspired by the structure of human photoreceptor topography [14], and has recently been shown to be very effective for improving nonlocal-means denoising algorithms [15]. It imposes a spatially-variant blur on the input window pixels, such that full detail is kept in the central section (*fovea*), while the peripheral parts are defocused by means of a convolution with a disk kernel, to remove fine details. The network, whose task is to classify the center pixel of the window, is then forced to disregard such peripheral fine details, which are most likely irrelevant, while still retaining the general structure of the window (context).

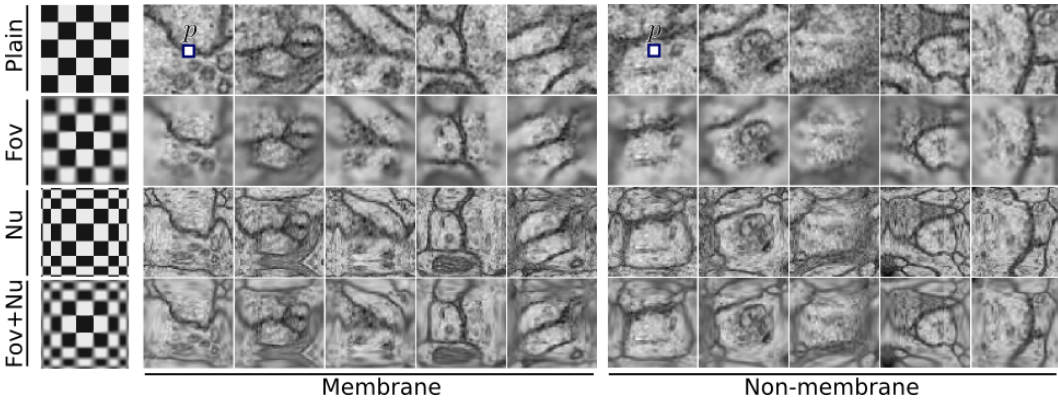

Figure 3: Input windows with $w = 65$, from the training set. First row shows original window (*Plain*); other rows show effects of foveation (*Fov*), nonuniform sampling (*Nu*), and both (*Fov+Nu*). Samples on the left and right correspond to instances of class *Membrane* and *Non-membrane*, respectively. The leftmost image illustrates how a checkerboard pattern is affected by such transformations.

*Nonuniform sampling* is motivated by the observation that (in this and other applications) larger window sizes $w$ generally result in significant performance improvements. However, a large $w$

results in much bigger networks, which take longer to train and, at least in theory, require larger amounts of training data to retain their generalization ability. With nonuniform sampling, image pixels are directly mapped to neurons only in the central part of the window; elsewhere, their source pixels are sampled with decreasing resolution as the distance from the window center increases. As a result, the image in the window is deformed in a fisheye-like fashion, and covers a larger area of the input image with fewer neurons.

Simultaneously applying both techniques is a way of exploiting data at multiple resolutions—fine at the center, coarse in the periphery of the window.

## 2.5 Averaging outputs of multiple networks

We observed that large networks with different architectures often exhibit significant output differences for many image parts, despite being trained on the same data. This suggests that these powerful and flexible classifiers exhibit relatively large variance but low bias. It is therefore reasonable to attempt to reduce such variance by averaging the calibrated outputs of several networks with different architectures.

This was experimentally verified. The submissions obtained by averaging the outputs of multiple large networks scored significantly better in all metrics than the single networks.

## 3 Experimental results

All experiments are performed on a computer with a Core i7 950 3.06GHz processor, 24GB of RAM, and four GTX 580 graphics cards. A GPU implementation [12] accelerates the forward propagation and back propagation routines by a factor of 50.

We validate our approach on the publicly-available dataset [9] provided by the organizers of the ISBI 2012 EM Segmentation Challenge [1], which represents two portions of the ventral nerve cord of a Drosophila larva. The dataset is composed by two $512 \times 512 \times 30$ stacks, one used for training, one for testing. Each stack covers a $2 \times 2 \times 1.5$ $\mu$m volume, with a resolution of $4 \times 4 \times 50$ nm/pixel. For the training stack, a manually annotated ground truth segmentation is provided. For the testing stack, the organizers obtained (but did not distribute) two manual segmentations by different expert neuroanatomists. One is used as ground truth, the other to evaluate the performance of a second human observer and provide a meaningful comparison for the algorithms' performance.

A segmentation of the testing stack is evaluated through an automated online system, which computes three error metrics in relation to the hidden ground truth:

**Rand error:** defined as $1 - F_{\mathrm{rand}}$, where $F_{\mathrm{rand}}$ represents the $F_1$ score of the *Rand index* [29], which measures the accuracy with which pixels are associated to their respective neurons.

**Warping error:** a segmentation metric designed to account for topological disagreements [19]; it accounts for the number of neuron splits and mergers required to obtain the candidate segmentation from ground truth.

**Pixel error:** defined as $1 - F_{\mathrm{pixel}}$, where $F_{\mathrm{pixel}}$ represents the $F_1$ score of pixel similarity.

The automated system accepts a stack of grayscale images, representing membrane probability values for each pixel; the stack is thresholded using 9 different threshold values, obtaining 9 binary stacks. For each of the stacks, the system computes the error measures above, and returns the minimum error.

Pixel error is clearly not a suitable indicator of segmentation quality in this context, and is reported mostly for reference. Rand and Warping error metrics have various strengths and weaknesses, without clear consensus in favor of any. The former tends to provide a more consistent measure but penalizes even slightly misplaced borders, which would not be problematic in most practical applications. The latter has a more intuitive interpretation, but completely disregards non-topological errors.

We train four networks N1, N2, N3 and N4, with slightly different architectures, and window sizes $w = 65$ (for N1, N2, N3) and $w = 95$ (for N4); all networks use foveation and nonuniform sampling,

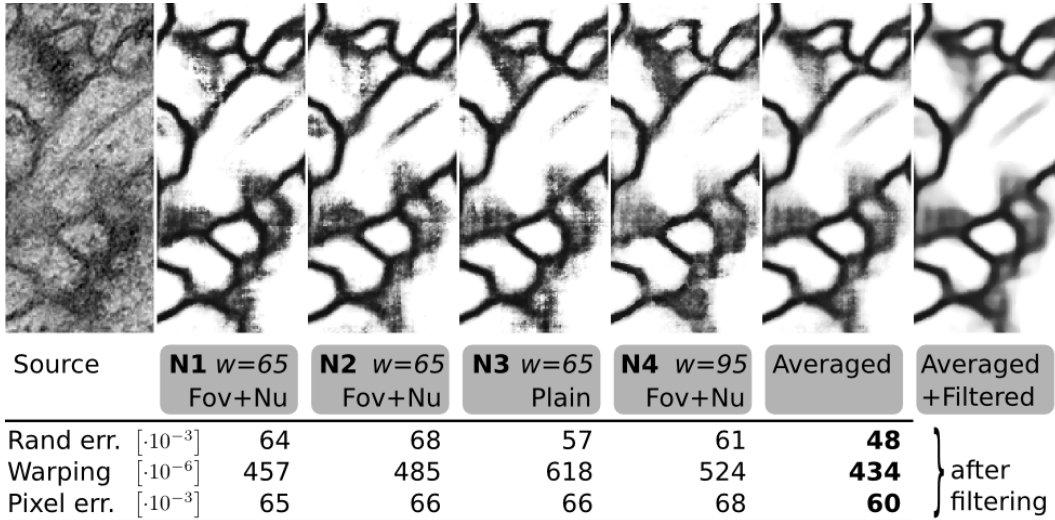

| | **N1** *w=65* Fov+Nu | **N2** *w=65* Fov+Nu | **N3** *w=65* Plain | **N4** *w=95* Fov+Nu | Averaged | Averaged +Filtered |
|---|---|---|---|---|---|---|
| Rand err. $[\cdot 10^{-3}]$ | 64 | 68 | 57 | 61 | **48** | |
| Warping $[\cdot 10^{-6}]$ | 457 | 485 | 618 | 524 | **434** | after |
| Pixel err. $[\cdot 10^{-3}]$ | 65 | 66 | 66 | 68 | **60** | filtering |

Figure 4: *Above*, from left to right: part of a source image from the test set; corresponding calibrated outputs of networks N1, N2, N3 and N4; average of such outputs; average after filtering. *Below*, the performance of each network, as well as the significantly better performance due to averaging their outputs. All results are computed after median filtering (see text).

except N3, which uses neither. As the input window size increases, the network depth also increases because we keep the convolutional filter sizes small. The architecture of N4 is the deepest, and is reported in Table 1.

Training time for one epoch varies from approximately 170 minutes for N1 ($w = 65$) to 340 minutes for N4 ($w = 95$). All nets are trained for 30 epochs, which leads to a total training time of several days. However, once networks are trained, application to new images is relatively fast: classifying the 8 million pixels comprising the whole testing stack takes 10 to 30 minutes on four GPUs. Such implementation is currently being further optimized (with foreseen speedups of one order of magnitude at least) in view of application to huge, terapixel-class datasets [6, 21].

Table 1: 11-layer architecture for network N4, $w = 95$.

| Layer | Type | Maps and neurons | Kernel size |
|---|---|---|---|
| 0 | input | 1 map of 95x95 neurons | |
| 1 | convolutional | 48 maps of 92x92 neurons | 4x4 |
| 2 | max pooling | 48 maps of 46x46 neurons | 2x2 |
| 3 | convolutional | 48 maps of 42x42 neurons | 5x5 |
| 4 | max pooling | 48 maps of 21x21 neurons | 2x2 |
| 5 | convolutional | 48 maps of 18x18 neurons | 4x4 |
| 6 | max pooling | 48 maps of 9x9 neurons | 2x2 |
| 7 | convolutional | 48 maps of 6x6 neurons | 4x4 |
| 8 | max pooling | 48 maps of 3x3 neurons | 2x2 |
| 9 | fully connected | 200 neurons | 1x1 |
| 10 | fully connected | 2 neurons | 1x1 |

The outputs of four such networks are shown in Figure 4, along with their performance after filtering. By averaging the outputs of all networks, results improve significantly. The final result for one slice of the test stack is shown in Figure 5.

Our results are compared to competing methods in Table 2.

Since our pure pixel classifier method aims at minimizing pixel error, Rand and warping errors are just minimized as a side-effect, but never explicitly accounted for during segmentation. In contrast, some competing segmentation approaches adopt different post-processing techniques directly opti-

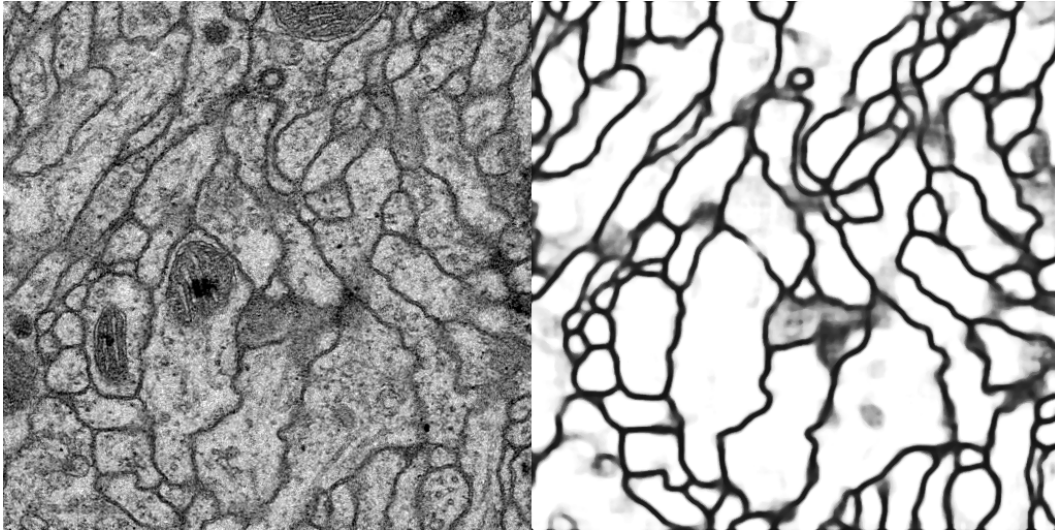

Figure 5: Left: slice 16 of the test stack. Right: corresponding output.

Table 2: Results of our approach and competing algorithms. For comparison, the first two rows report the performance of the second human observer and of a simple thresholding approach.

| Group | Rand error [$\cdot 10^{-3}$] | Warping error [$\cdot 10^{-6}$] | Pixel error [$\cdot 10^{-3}$] |
|---|---|---|---|
| *Second Human Observer* | *27* | *344* | *67* |
| *Simple Thresholding* | *445* | *15522* | *222* |
| Our approach | **48** | **434** | **60** |
| Laptev et al. [24] (1) | 65 | 556 | 83 |
| Laptev et al. [24] (2) | 70 | 525 | 79 |
| Sumbul et al. | 76 | 646 | 65 |
| Liu et al. [26] (1) | 84 | 1602 | 134 |
| Kaynig et al. [23] | 84 | 1124 | 157 |
| Liu et al. [26] (2) | 89 | 1134 | 78 |
| Kamentsky et al. [20] | 90 | 1512 | 100 |
| Burget et al. [8] | 139 | 2641 | 102 |
| Tan et al. [36] | 153 | 685 | 88 |
| Bas et al. [4] | 162 | 1613 | 109 |
| Iftikhar et al. [18] | 230 | 16156 | 150 |

mizing the rand error. Nevertheless, their results are inferior. But such post-processing techniques—which unlike our general classifier are specific to this particular problem—could be successfully applied to finetune our outputs, further improving results. Preliminary results in this direction are encouraging: the problem-specific postprocessing techniques in [20] and [24], operating on our segmentation, reduce the Rand error to measure to $36 \cdot 10^{-3}$ and $32 \cdot 10^{-3}$, respectively. Further research along these lines is planned for the near future.

## 4   Discussion and conclusions

The main strength of our approach to neuronal membrane segmentation in EM images lies in a deep and wide neural network trained by online back-propagation to become a very powerful pixel classifier with superhuman pixel-error rate, made possible by an optimized GPU implementation more than 50 times faster than equivalent code on standard microprocessors.

Our approach outperforms all other approaches in the competition, despite not even being tailored to this particular segmentation task. Instead, the DNN acts as a generic image classifier, using raw pixel intensities as inputs, without ad-hoc post-processing. This opens interesting perspectives on applying similar techniques to other biomedical image segmentation tasks.

### Acknowledgments

This work was partially supported by the *Supervised Deep / Recurrent Nets* SNF grant, Project Code 140399.

## References

[1] Segmentation of neuronal structures in EM stacks challenge - ISBI 2012. `http://tinyurl.com/d2fgh7g`.

[2] The Open Connectome Project. `http://openconnectomeproject.org`.

[3] R. Achanta, A. Shaji, K. Smith, A. Lucchi, P. Fua, and S. Süsstrunk. Slic superpixels. *Technical Report 149300 EPFL*, (June), 2010.

[4] Erhan Bas, Mustafa G. Uzunbas, Dimitris Metaxas, and Eugene Myers. Contextual grouping in a concept: a multistage decision strategy for EM segmentation. In *Proc. of ISBI 2012 EM Segmentation Challenge*.

[5] Sven Behnke. *Hierarchical Neural Networks for Image Interpretation*, volume 2766 of *Lecture Notes in Computer Science*. Springer, 2003.

[6] Davi D. Bock, Wei-Chung A. Lee, Aaron M. Kerlin, Mark L. Andermann, Greg Hood, Arthur W. Wetzel, Sergey Yurgenson, Edward R. Soucy, Hyon S. Kim, and R. Clay Reid. Network anatomy and in vivo physiology of visual cortical neurons. *Nature*, 471(7337):177–182, 2011.

[7] Y. Boykov, O. Veksler, and R. Zabih. Fast approximate energy minimization via graph cuts. *Pattern Analysis and Machine Intelligence, IEEE Transactions on*, 23(11):1222–1239, 2001.

[8] Radim Burget, Vaclav Uher, and Jan Masek. Trainable Segmentation Based on Local-level and Segment-level Feature Extraction. In *Proc. of ISBI 2012 EM Segmentation Challenge*.

[9] Albert Cardona, Stephan Saalfeld, Stephan Preibisch, Benjamin Schmid, Anchi Cheng, Jim Pulokas, Pavel Tomancak, and Volker Hartenstein. An integrated micro- and macroarchitectural analysis of the drosophila brain by computer-assisted serial section electron microscopy. *PLoS Biol*, 8(10):e1000502, 10 2010.

[10] Dan Claudiu Ciresan, Ueli Meier, Luca Maria Gambardella, and Jürgen Schmidhuber. Deep, big, simple neural nets for handwritten digit recognition. *Neural Computation*, 22(12):3207–3220, 2010.

[11] Dan Claudiu Ciresan, Ueli Meier, Luca Maria Gambardella, and Jürgen Schmidhuber. Convolutional neural network committees for handwritten character classification. In *International Conference on Document Analysis and Recognition*, pages 1250–1254, 2011.

[12] Dan Claudiu Ciresan, Ueli Meier, Jonathan Masci, Luca Maria Gambardella, and Jürgen Schmidhuber. Flexible, high performance convolutional neural networks for image classification. In *International Joint Conference on Artificial Intelligence*, pages 1237–1242, 2011.

[13] Dan Claudiu Ciresan, Ueli Meier, and Jürgen Schmidhuber. Multi-column deep neural networks for image classification. In *Computer Vision and Pattern Recognition*, pages 3642–3649, 2012.

[14] C.A. Curcio, K.R. Sloan, R.E. Kalina, and A.E. Hendrickson. Human photoreceptor topography. *The Journal of comparative neurology*, 292(4):497–523, 1990.

[15] A. Foi and G. Boracchi. Foveated self-similarity in nonlocal image filtering. In *Proceedings of SPIE*, volume 8291, page 829110, 2012.

[16] Kunihiko Fukushima. Neocognitron: A self-organizing neural network for a mechanism of pattern recognition unaffected by shift in position. *Biological Cybernetics*, 36(4):193–202, 1980.

[17] G. González, F. Fleurety, and P. Fua. Learning rotational features for filament detection. In *Computer Vision and Pattern Recognition, 2009. CVPR 2009. IEEE Conference on*, pages 1582–1589. IEEE, 2009.

[18] Saadia Iftikhar and Afzal Godil. The Detection of Neuronal Structures using a Patch-based Multi-features and Support Vector Machines Learning Algorithm. In *Proc. of ISBI 2012 EM Segmentation Challenge*.

[19] Viren Jain, Benjamin Bollmann, Mark Richardson, Daniel R. Berger, Moritz Helmstaedter, Kevin L. Briggman, Winfried Denk, Jared B. Bowden, John M. Mendenhall, Wickliffe C. Abraham, Kristen M. Harris, N. Kasthuri, Ken J. Hayworth, Richard Schalek, Juan Carlos Tapia, Jeff W. Lichtman, and H. Sebastian Seung. Boundary Learning by Optimization with Topological Constraints. In *CVPR*, pages 2488–2495. IEEE, 2010.

[20] Lee Kamentsky. Segmentation of EM images of neuronal structures using CellProfiler. In *Proc. of ISBI 2012 EM Segmentation Challenge*.

[21] Bobby Kasthuri. Mouse Visual Cortex Dataset in the Open Connectome Project. http://openconnectomeproject.org/Kasthuri11/.

[22] V. Kaynig, T. Fuchs, and J. Buhmann. Geometrical consistent 3D tracing of neuronal processes in ssTEM data. *Medical Image Computing and Computer-Assisted Intervention–MICCAI 2010*, pages 209–216, 2010.

[23] V. Kaynig, T. Fuchs, and J.M. Buhmann. Neuron geometry extraction by perceptual grouping in sstem images. In *Computer Vision and Pattern Recognition (CVPR), 2010 IEEE Conference on*, pages 2902–2909. IEEE, 2010.

[24] Dmitry Laptev, Alexander Vezhnevets, Sarvesh Dwivedi, and Joachim Buhmann. Segmentation of Neuronal Structures in EM stacks. In *Proc. of ISBI 2012 EM Segmentation Challenge*.

[25] Y. LeCun, L. Bottou, Y. Bengio, and P. Haffner. Gradient-based learning applied to document recognition. *Proceedings of the IEEE*, 86(11):2278–2324, November 1998.

[26] Ting Liu, Mojtaba Seyedhosseini, Elizabeth Jurrus, and Tolga Tasdizen. Neuron Segmentation in EM Images using Series of Classifiers and Watershed Tree. In *Proc. of ISBI 2012 EM Segmentation Challenge*.

[27] A. Lucchi, K. Smith, R. Achanta, G. Knott, and P. Fua. Supervoxel-Based Segmentation of Mitochondria in EM Image Stacks With Learned Shape Features. *Medical Imaging, IEEE Transactions on*, (99):1–1, 2012.

[28] A. Lucchi, K. Smith, R. Achanta, V. Lepetit, and P. Fua. A fully automated approach to segmentation of irregularly shaped cellular structures in EM images. *Medical Image Computing and Computer-Assisted Intervention–MICCAI 2010*, pages 463–471, 2010.

[29] W.M. Rand. Objective criteria for the evaluation of clustering methods. *Journal of the American Statistical association*, 66(336):846–850, 1971.

[30] Maximiliam Riesenhuber and Tomaso Poggio. Hierarchical models of object recognition in cortex. *Nat. Neurosci.*, 2(11):1019–1025, 1999.

[31] Dominik Scherer, Adreas Müller, and Sven Behnke. Evaluation of pooling operations in convolutional architectures for object recognition. In *International Conference on Artificial Neural Networks*, 2010.

[32] Thomas Serre, Lior Wolf, and Tomaso Poggio. Object recognition with features inspired by visual cortex. In *Proc. of Computer Vision and Pattern Recognition Conference*, 2005.

[33] Patrice Y. Simard, Dave. Steinkraus, and John C. Platt. Best practices for convolutional neural networks applied to visual document analysis. In *Seventh International Conference on Document Analysis and Recognition*, pages 958–963, 2003.

[34] K. Smith, A. Carleton, and V. Lepetit. Fast ray features for learning irregular shapes. In *Computer Vision, 2009 IEEE 12th International Conference on*, pages 397–404. IEEE, 2009.

[35] Daniel Strigl, Klaus Kofler, and Stefan Podlipnig. Performance and scalability of GPU-based convolutional neural networks. In *18th Euromicro Conference on Parallel, Distributed, and Network-Based Processing*, 2010.

[36] Xiao Tan and Changming Sun. Membrane extraction using two-step classification and post-processing. In *Proc. of ISBI 2012 EM Segmentation Challenge*.

